# Signal-to-Noise Ratio Analysis of Policy Gradient Algorithms

**John W. Roberts and Russ Tedrake**
Computer Science and
Artificial Intelligence Laboratory
Massachusetts Institute of Technology
Cambridge, MA 02139

## Abstract

Policy gradient (PG) reinforcement learning algorithms have strong (local) convergence guarantees, but their learning performance is typically limited by a large variance in the estimate of the gradient. In this paper, we formulate the variance reduction problem by describing a signal-to-noise ratio (SNR) for policy gradient algorithms, and evaluate this SNR carefully for the popular Weight Perturbation (WP) algorithm. We confirm that SNR is a good predictor of long-term learning performance, and that in our episodic formulation, the cost-to-go function is indeed the optimal baseline. We then propose two modifications to traditional model-free policy gradient algorithms in order to optimize the SNR. First, we examine WP using anisotropic sampling distributions, which introduces a bias into the update but increases the SNR; this bias can be interpreted as following the natural gradient of the cost function. Second, we show that non-Gaussian distributions can also increase the SNR, and argue that the optimal isotropic distribution is a 'shell' distribution with a constant magnitude and uniform distribution in direction. We demonstrate that both modifications produce substantial improvements in learning performance in challenging policy gradient experiments.

## 1 Introduction

Model-free policy gradient algorithms allow for the optimization of control policies on systems which are impractical to model effectively, whether due to cost, complexity or uncertainty in the very structure and dynamics of the system (Kohl & Stone, 2004; Tedrake et al., 2004). However, these algorithms often suffer from high variance and relatively slow convergence times (Greensmith et al., 2004). As the same systems on which one wishes to use these algorithms tend to have a high cost of policy evaluation, much work has been done on maximizing the policy improvement from any individual evaluation (Meuleau et al., 2000; Williams et al., 2006). Techniques such as Natural Gradient (Amari, 1998; Peters et al., 2003a) and GPOMDP (Baxter & Bartlett, 2001) have become popular through their ability to match the performance gains of more basic model-free policy gradient algorithms while using fewer policy evaluations.

As practitioners of policy gradient algorithms in complicated mechanical systems, our group has a vested interest in making practical and substantial improvements to the performance of these algorithms. Variance reduction, in itself, is not a sufficient metric for optimizing the performance of PG algorithms - of greater significance is the magnitude of the variance relative to the magnitude of the gradient update. Here we formulate a signal-to-noise ratio (SNR) which facilitates simple and fast evaluations of a PG algorithm's average performance, and facilitates algorithmic performance improvements. Though the SNR does not capture all facets of a policy gradient algorithm's capability to learn, we show that achieving a high SNR will often result in a superior convergence rate with less violent variations in the policy.

Through a close analysis of the SNR, and the means by which it is maximized, we find several modifications to traditional model-free policy gradient updates that improve learning performance. The first of these is the reshaping of distributions such that they are different on different parameters, a modification which introduces a bias to the update. We show that this reshaping can improve performance, and that the introduced bias results in following the natural gradient of the cost function, rather than the true point gradient. The second improvement is the use of non-Gaussian distributions for sampling, and through the SNR we find a simple distribution which improves performance without increasing the complexity of implementation.

## 2 The weight perturbation update

Consider minimizing a scalar function $J(\vec{w})$ with respect to the parameters $\vec{w}$ (note that it is possible that $J(\vec{w})$ is a long-term cost and results from running a system with the parameters $\vec{w}$ until conclusion). The weight perturbation algorithm (Jabri & Flower, 1992) performs this minimization with the update:

$$\Delta \vec{w} = -\eta \left( J(\vec{w} + \vec{z}) - J(\vec{w}) \right) \vec{z}, \tag{1}$$

where the components of the 'perturbation', $\vec{z}$, are drawn independently from a mean-zero distribution, and $\eta$ is a positive scalar controlling the magnitude of the update (the "learning rate"). Performing a first-order Taylor expansion of $J(\vec{w} + \vec{z})$ yields:

$$\Delta \vec{w} = -\eta \left( J(\vec{w}) + \sum_i \frac{\partial J}{\partial \vec{w}_i} z_i - J(\vec{w}) \right) \vec{z} = -\eta \sum_i \frac{\partial J}{\partial \vec{w}_i} z_i \cdot \vec{z}. \tag{2}$$

In expectation, this becomes the gradient times a (diagonal) covariance matrix, and reduces to

$$E[\Delta \vec{w}] = -\eta \sigma^2 \frac{\partial J}{\partial \vec{w}}, \tag{3}$$

an unbiased estimate of the gradient, scaled by the learning rate and $\sigma^2$, the variance of the perturbation. However, this unbiasedness comes with a very high variance, as the direction of an update is uniformly distributed. It is only the fact that updates near the direction of the true gradient have a larger magnitude than do those nearly perpendicular to the gradient that allows for the true gradient to be achieved in expectation. Note also that all samples parallel to the gradient are equally useful, whether they be in the same or opposite direction, as the sign does not affect the resulting update.

The WP algorithm is one of the simplest examples of a policy gradient reinforcement learning algorithm, and thus is well suited for analysis. In the special case when $\vec{z}$ is drawn from a Gaussian distribution, weight perturbation can be interpreted as a REINFORCE update(Williams, 1992).

## 3 SNR for policy gradient algorithms

The SNR is the expected power of the signal (update in the direction of the true gradient) divided by the expected power of the noise (update perpendicular to the true gradient). Taking care to ensure that the magnitude of the true gradient does not effect the SNR, we have:

$$\text{SNR} = \frac{E\left[ \Delta \vec{w}_\parallel^T \Delta \vec{w}_\parallel \right]}{E\left[ \Delta \vec{w}_\perp^T \Delta \vec{w}_\perp \right]}, \tag{4}$$

$$\Delta \vec{w}_\parallel = \left( \Delta \vec{w}^T \frac{\vec{J_w}}{\left\| \vec{J_w} \right\|} \right) \frac{\vec{J_w}}{\left\| \vec{J_w} \right\|}, \quad \Delta \vec{w}_\perp = \Delta \vec{w} - \vec{w}_\parallel, \tag{5}$$

and using $\vec{J_w}(\vec{w}_0) = \left. \frac{\partial J(\vec{w})}{\partial \vec{w}} \right|_{(\vec{w} = \vec{w}_0)}$ for convenience.

Intuitively, this expression measures how large a proportion of the update is "useful". If the update is purely in the direction of the gradient the SNR would be infinite, while if the update moved perpendicular to the true gradient, it would be zero. As such, all else being equal, a higher SNR should generally perform as well or better than a lower SNR, and result in less violent swings in cost and policy for the same improvement in performance.

## 3.1 Weight perturbation with Gaussian distributions

Evaluating the SNR for the WP update in Equation 1 with a deterministic $J(\vec{w})$ and $\vec{z}$ drawn from a Gaussian distribution yields a surprisingly simple result. If one first considers the numerator:

$$
\begin{aligned}
E\left[\Delta \vec{w}_\parallel^T \Delta \vec{w}_\parallel\right] &= E\left[\frac{\eta^2}{\left\|\vec{J}_w\right\|^4}\left(\sum_{i,j} J_{w_i} J_{w_j} z_i z_j\right)\vec{J}_w^T \cdot \left(\sum_{k,p} J_{w_k} J_{w_p} z_k z_p\right)\vec{J}_w\right] \\
&= E\left[\frac{\eta^2}{\left\|\vec{J}_w\right\|^2}\sum_{i,j,k,p} J_{w_i} J_{w_j} J_{w_k} J_{w_p} z_i z_j z_k z_p\right] = Q,
\end{aligned}
\tag{6}
$$

where we have named this term $Q$ for convenience as it occurs several times in the expansion of the SNR. We now expand the denominator as follows:

$$
E\left[\Delta \vec{w}_\perp^T \Delta \vec{w}_\perp\right] = E\left[\Delta \vec{w}^T \Delta \vec{w} - 2\Delta \vec{w}_\parallel^T(\Delta \vec{w}_\parallel + \Delta \vec{w}_\perp) + \Delta \vec{w}_\parallel^T \Delta \vec{w}_\parallel\right] = E\left[\Delta \vec{w}^T \Delta \vec{w}\right] - 2Q + Q
\tag{7}
$$

Substituting Equation (1) into Equation (7) and simplifying results in:

$$
E\left[\Delta \vec{w}_\perp^T \Delta \vec{w}_\perp\right] = \frac{\eta^2}{\left\|\vec{J}_w\right\|^2}E\left[\sum_{i,j,k} J_{w_i} J_{w_j} z_i z_j z_k^2\right] - Q.
\tag{8}
$$

We now assume that each component $z_i$ is drawn from a Gaussian distribution with variance $\sigma^2$. Taking the expected value, it may be further simplified to:

$$
Q = \frac{\eta^2}{\left\|\vec{J}_w\right\|^4}\left(3\sigma^4 \sum_i J_{w_i}^4 + 3\sigma^4 \sum_i J_{w_i}^2 \sum_{j \neq i} J_{w_j}^2\right) = \frac{3\sigma^4}{\left\|\vec{J}_w\right\|^4}\sum_{i,j} J_{w_i}^2 J_{w_j}^2 = 3\sigma^4,
\tag{9}
$$

$$
E\left[\Delta \vec{w}_\perp^T \Delta \vec{w}_\perp\right] = \frac{\eta^2 \sigma^4}{\left\|\vec{J}_w\right\|^2}\left(2\sum_i J_{w_i}^2 + \sum_{i,j} J_{w_i}^2\right) - Q = \sigma^4(2+N) - 3\sigma^4 = \sigma^4(N-1),
\tag{10}
$$

where $N$ is the number of parameters. Canceling $\sigma$ results in:

$$
\text{SNR} = \frac{3}{N-1}.
\tag{11}
$$

Thus, for small noises and constant $\sigma$ the SNR and the parameter number have a simple inverse relationship. This is a particularly concise model for performance scaling in PG algorithms.

## 3.2 Relationship of the SNR to learning performance

To evaluate the degree to which the SNR is correlated with actual learning performance, we ran a number of experiments on a simple quadratic bowl cost function, which may be written as:

$$
J(\vec{w}) = \vec{w}^T A \vec{w},
\tag{12}
$$

where the optimal is always at the point $\vec{0}$. The SNR suggests a simple inverse relationship between the number of parameters and the learning performance. To evelute this claim we performed three tests: 1) true gradient descent on the identity cost function ($A$ set to the identity matrix) as a benchmark, 2) WP on the identity cost function and 3) WP on 150 randomly generated cost functions (each component drawn from a Gaussian distribution), all of the form given in Equation (12), and for values of $N$ between 2 and 10. For each trial $\vec{w}$ was intially set to be $\vec{1}$. As can be seen in Figure 1a, both the SNR and the reduction in cost after running WP for 100 iterations decrease monotonically as the number of parameters $N$ increases. The fact that this occurs in the case of randomly generated cost functions demonstrates that this effect is not related to the simple form of the identity cost function, but is in fact related to the number of dimensions.

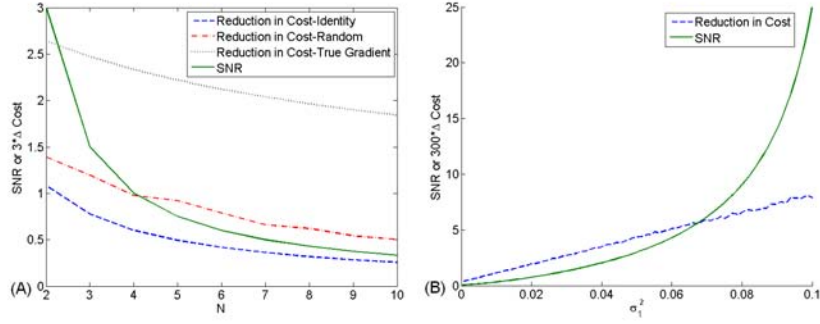

Figure 1: Two comparisons of SNR and learning performance: (A) Relationship as dimension $N$ is increased (Section 3.2). The curves are 15,000 averaged runs, each run 100 iterations. For randomly generated cost functions, 150 $A$ matrices were tested. True gradient descent was run on the identity cost function. The SNR for each case was computed in with Equation (11). (B) Relationship as Gaussian is reshaped by changing variances for case of 2D anisotropic cost function(ratio of gradients in different directions is 5) cost function (Section 4.1.1). The constraint $\sigma_1^2 + \sigma_2^2 = 0.1$ is imposed, while $\sigma_1^2$ is between 0 and .1. For each value of $\sigma_1$ 15,000 updates were averaged to produce the curve plotted. The plot shows that variances which increase the SNR also improve the performance of the update.

### 3.3   SNR with parameter-independent additive noise

In many real world systems, the evaluation of the cost $J(\vec{w})$ is not deterministic, a property which can significantly affect learning performance. In this section we investigate how additive 'noise' in the function evaluation affects the analytical expression for the SNR. We demonstrate that for very high noise WP begins to behave like a random walk, and we find in the SNR the motivation for an improvement in the WP algorithm that will be examined in Section 4.2.

Consider modifying the update seen in Equation (1) to allow for a parameter-independent additive noise term $v$ and a more general baseline $b(\vec{w})$, and again perform the Taylor expansion. Writing the update with these terms gives:

$$\Delta \vec{w} = -\eta \left( J(\vec{w}) + \sum_i J_{w_i} z_i - b(\vec{w}) + v \right) \vec{z} = -\eta \left( \sum_i J_{w_i} z_i + \xi(\vec{w}) \right) \vec{z}. \qquad (13)$$

where we have combined the terms $J(\vec{w})$, $b(\vec{w})$ and $v$ into a single random variable $\xi(\vec{w})$. The new variable $\xi(\vec{w})$ has two important properties: its mean can be controlled through the value of $b(\vec{w})$, and its distribution is independent of parameters $\vec{w}$, thus $\xi(\vec{w})$ is independent of all the $z_i$.

We now essentially repeat the calculation seen in Section 3.1, with the small modification of including the noise term. When we again assume independent $z_i$, each drawn from identical Gaussian distributions with standard deviation $\sigma$, we obtain the expression:

$$\text{SNR} = \frac{\phi + 3}{(N-1)(\phi+1)}, \quad \phi = \frac{(J(\vec{w}) - b(\vec{w}))^2 + \sigma_v^2}{\sigma^2 \|\vec{J_w}\|^2} \qquad (14)$$

where $\sigma_v$ is the standard deviation of the noise $v$ and we have termed the error component $\phi$. This expression depends upon the fact that the noise $v$ is mean-zero and independent of the parameters, although as stated earlier, the assumption that $v$ is mean-zero is not limiting. It is clear that in the limit of small $\phi$ the expression reduces to that seen in Equation (11), while in the limit of very large $\phi$ it becomes the expression for the SNR of a random walk (see Section 3.4). This expression makes it clear that minimizing $\phi$ is desirable, a result that suggests two things: (1) the optimal baseline (from the perspective of the SNR) is the value function (i.e. $b^*(\vec{w}) = J(\vec{w})$) and (2) higher values of $\sigma$ are desirable, as they reduce $\phi$ by increasing the size of its denominator. However, there is clearly a limit on the size of $\sigma$ due to higher order terms in the Taylor expansion; very large $\sigma$ will result in samples which do not represent the local gradient. Thus, in the case of noisy measurements, there is some optimal sampling distance that is as large as possible without resulting in poor sampling of the local gradient. This is explored in Section 4.2.1.

### 3.4 SNR of a Random Walk

Due to the fact that the update is squared in the SNR, only its degree of parallelity to the true gradient is relevant, not its direction. In the case of WP on a deterministic function, this is not a concern as the update is always within $90°$ of the gradient, and thus the parallel component is always in the correct direction. For a system with noise, however, components of the update parallel to the gradient can in fact be in the incorrect direction, contributing to the SNR even though they do not actually result in learning. This effect only becomes significant when the noise is particularly large, and reaches its extreme in the case of a true random walk (a strong bias in the "wrong" direction is in fact a good update with an incorrect sign). If one considers moving by a vector drawn from a multivariate Gaussian distribution without any correlation to the cost function, the SNR is particularly easy to compute, taking the form:

$$\mathrm{SNR} = \frac{\frac{1}{\|\vec{J_w}\|^4} \sum_i J_{w_i} z_i \vec{J_w}^T \sum_j J_{w_j} z_j \vec{J_w}}{(\vec{z} - \frac{1}{\|\vec{J_w}\|^2} \sum_i J_{w_i} z_i \vec{J_w})^T (\vec{z} - \frac{1}{\|\vec{J_w}\|^2} \sum_i J_{w_i} z_i \vec{J_w})} = \frac{\sigma^2}{N\sigma^2 - 2\sigma^2 + \sigma^2} = \frac{1}{N-1}$$

(15)

As was discussed in Section 3.3, this value of the SNR is the limiting case of very high measurement noise, a situation which will in fact produce a random walk.

## 4 Applications of SNR

### 4.1 Reshaping the Gaussian Distribution

Consider a generalized WP algorithm, in which we allow each component $z_i$ to be drawn independently from separate mean-zero distributions. Returning to the derivation in Section 3.1, we no longer assume each $z_i$ is drawn from an identical distribution, but rather associate each with its own $\sigma_i$ (the vector of the $\sigma_i$ will be referred to as $\vec{\sigma}$). Removing this assumption results in the SNR:

$$\mathrm{SNR}(\vec{\sigma}, \vec{J_w}) = \left[ \frac{\left\|\vec{J_w}\right\|^2 \left( 2\sum_i J_{w_i}{}^2 \sigma_i^4 + \sum_{i,j} J_{w_i}{}^2 \sigma_i^2 \sigma_j^2 \right)}{3 \sum_{i,j} J_{w_i}{}^2 \sigma_i^2 J_{w_j}{}^2 \sigma_j^2} - 1 \right]^{-1}.$$

(16)

An important property of this SNR is that it depends only upon the direction of $\vec{J_w}$ and the relative magnitude of the $\sigma_i$ (as opposed to parameters such as the learning rate $\eta$ and the absolute magnitudes $\|\vec{\sigma}\|$ and $\|\vec{J_w}\|$).

#### 4.1.1 Effect of reshaping on performance

While the absolute magnitudes of the variance and true gradient do not affect the SNR given in Equation (16), the relative magnitudes of the different $\sigma_i$ and their relationship to the true gradient *can* affect it. To study this property, we investigate a cost function with a significant degree of anisotropy. Using a cost function of the form given in Equation (12) and $N = 2$, we choose an $A$ matrix whose first diagonal component is five times that of the second. We then investigate a series of possible variances $\sigma_1^2$ and $\sigma_2^2$ constrained such that their sum is a constant ($\sigma_1^2 + \sigma_2^2 = C$). We observe the performance of the first update (rather than the full trial) as the true gradient can vary significantly over the course of a trial, thereby having major effects on the SNR even as the variances are unchanged. As is clear in Figure 1b, as the SNR is increased through the choice of variances the performance of this update is improved. The variation of the SNR is much more significant than the change in performance, however this is not surprising as the SNR is infinite if the update is exactly along the correct direction, while the improvement from this update will eventually saturate.

### 4.1.2  Demonstration in simulation

The improved performance of the previous section suggests the possibility of a modification to the WP algorithm in which an estimate of the true gradient is used before each update to select new variances which are more likely to learn effectively. Changing the shape of the distribution does add a bias to the update direction, but the resulting biased update is in fact descending the natural gradient of the cost function. To make use of this opportunity, some knowledge of the likely gradient direction is required. This knowledge can be provided via a momentum estimate (an average of previous updates) or through an inaccurate model that is able to capture some facets of the geometry of the cost function. With this estimated gradient the expression given in Equation (16) can be optimized over the $\sigma_i$ numerically using a method such as Sequential Quadratic Programming (SQP). Care must be taken to avoid converging to very narrow distributions (e.g. placing some small minimum noise on all parameters regardless of the optimization), but ultimately this reshaping of the Gaussian can provide real performance benefits.

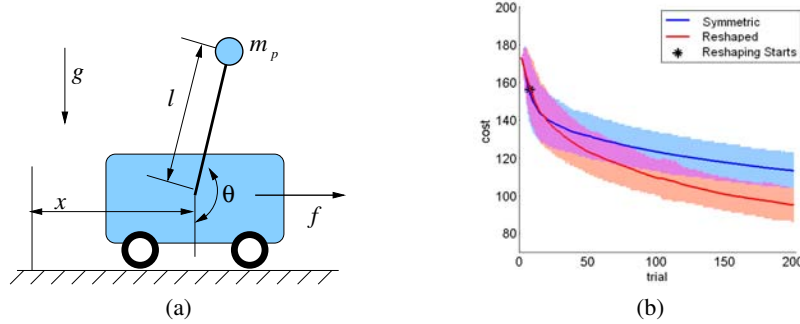

(a)                                                                                          (b)

Figure 2: (a) The cart-pole system. The task is to apply a horizontal force $f$ to the cart such that the pole swings to the vertical position. (b) The average of 200 curves showing reduction in cost versus trial number for both a symmetric Gaussian distribution and a distribution reshaped using the SNR. The blue shaded region marks the area within one standard deviation for a symmetric Gaussian distribution, the red region marks one standard deviation for the reshaped distribution and the purple is within one standard deviation of both. The reshaping began on the eighth trial to give time for the momentum-based gradient estimate to stabilize.

To demonstrate the improvement in convergence time this reshaping can achieve, weight perturbation was used to develop a barycentric feedback policy for the cart-pole swingup task, where the cost was defined as a weighted sum of the actuation used and the squared distance from the upright position. A gradient estimate was obtained through averaging previous updates, and SQP was used to optimize the SNR prior to each trial. Figure 2 demonstrates the superior performance of the reshaped distribution over a symmetric Guassian using the same total variance (i.e. the traces of the covariance matrices for both distributions were the same).

### 4.1.3  WP with Gaussian distributions follow the natural gradient

The natural gradient for a policy that samples with a mean-zero Gaussian of covariance $\Sigma$ may be written (see (Peters et al., 2003b)):

$$\tilde{\vec{J}}_w = F^{-1}\vec{J}_w, \quad F = E_{\pi(\vec{\xi};\vec{w})}\left[\frac{\partial \log \pi(\vec{\xi};\vec{w})}{\partial w_i}\frac{\partial \log \pi(\vec{\xi};\vec{w})}{\partial w_j}\right]. \tag{17}$$

where $F$ is the Fisher Information matrix, $\pi$ is the sampling distribution, and $\vec{\xi} = \vec{w} + \vec{z}$. Using the Gaussian form of the sampling, $F$ may be evaluated easily, and becomes as $\Sigma^{-1}$, thus:

$$\tilde{\vec{J}}_w = \Sigma \, \vec{J}_w. \tag{18}$$

This is true for all mean-zero multivariate Gaussian distributions, thus the biased update, while no longer following the local point gradient, does follow the natural gradient. It is important to note that the natural gradient is a function of the shape of the sampling distribution, and it is because of this that all sampling distributions of this form can follow the natural gradient.

## 4.2 Non-Gaussian Distributions

The analysis in Section 3.3 suggests that for a function with noisy measurements there is an optimal sampling distance which depends upon the local noise and gradient as well as the strength of higher-order terms in that region. For a two-dimensional cost function of the form given in Equation (12), Figure 3 shows the SNR's dependence upon the radius of the shell distribution (i.e. the magnitude of the sampling). For various levels of additive mean-zero noise the SNR was computed for a distribution uniform in angle and fixed in its distance from the mean (this distance is the "sampling magnitude"). The fact that there is a unique maximum for each case suggests the possibility of sampling *only* at that maximal magnitude, rather than over all magnitudes as is done with a Gaussian, and thus improving SNR and performance. While determining the exact magnitude of maximum SNR may be impractical, choosing a distribution with uniformly distributed direction and a constant magnitude close to this optimal value, performance can be improved. This idea was tested on the benchmark proposed in (Riedmiller et al., 2007), where comparisons showed it was able to learn at rates similar to optimized RPROP from reasonable initial policies, and was capable of learning from a zero initial policy.

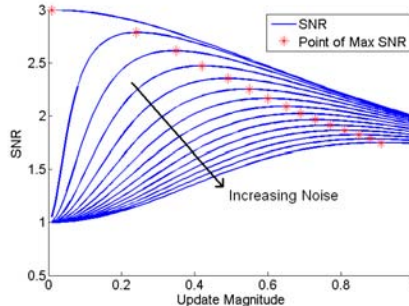

Figure 3: SNR vs. update magnitude for a 2D quadratic cost function. Mean-zero measurement noise is included with variances from 0 to .65. As the noise is increased, the sampling magnitude producing the maximum SNR is larger and the SNR achieved is lower. Note that the highest SNR achieved is for the smallest sampling magnitude with no noise where it approaches the theoretical value (for 2D) of 3. Also note that for small sampling magnitudes and large noises the SNR approaches the random walk value.

### 4.2.1 Experimental Demonstration

To provide compelling evidence of improved performance, the shell distribution was implemented on a laboratory experimental system with actuator limitations and innate stochasticity. We have recently been exploring the use of PG algorithms in an incredibly difficult and exciting control domain -fluid dynamics - and as such applied the shell distribution to a fluid dynamical system. Specifically, we applied learning to a system used to sudy the dynamics of flapping flight via a wing submerged in water (see Figure 4 for a description of the system (Vandenberghe et al., 2004)). The task is to determine the vertical motion producing the highest ratio of rotational displacement to energy input. Model-free methods are particularly exciting in this domain because direct numerical simulation can take days(Shelley et al., 2005) - in contrast optimizationg on the experimental physical flapping wing can be done in real-time, at the cost of dealing with noise in the evaluation of the cost function; success here would be enabling for experimental fluid dynamics. We explored the idea of using a "shell" distribution to improve the performance of our PG learning on this real-world system.

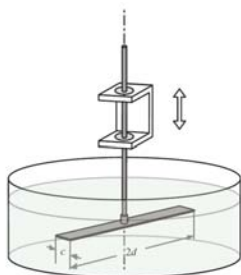

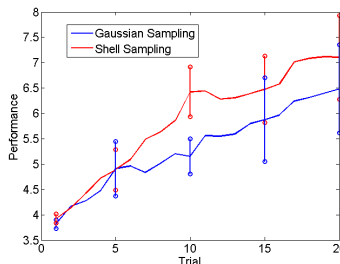

(a)  (b)

Figure 4: (a) Schematic of the flapping setup. The plate rotates freely about its vertical axis, while the vertical motion is prescribed by the learnt policy. This vertical motion is coupled with the plate's rotation through hydrodynamic effects. (b) 5 averaged runs on the flapping plate using Gaussian or Shell distributions for sampling. The error bars represent one standard deviation in the performance of different runs at that trial.

Representing the vertical position as a function of time with a 13-point periodic cubic spline, a 5D space was searched (points 1, 7 and 13 were fixed at zero, while points 2 and 8, 3 and 9 etc. were set to equal and opposite values determined by the control parameters). Beginning with a smoothed square wave, WP was run for 20 updates using shell distributions and Gaussians. Both forms of distributions were run 5 times and averaged to produce the curves in Figure 4. The sampling magnitude of the shell distribution was set to be the expected value of the length of a sample from the Gaussian distribution, while all other parameters were set equal. With optimized sampling, we acquired locally optimal policies in as little as 15 minutes, with repeated optimizations from very different initial policies converging to the same waveform. The result deepened our understanding of this fluid system and suggests promising applications to other fluid systems of similar complexity.

## 5 Conclusion

In this paper we present an expression for the SNR of PG algorithms, and looked in detail at the common case of WP. This expression gives us a quantitative means of evaluating the expected performance of a PG algorithm, although the SNR does not completely capture an algorithm's capacity to learn. SNR analysis revealed two distinct mechanisms for improving the WP update - perturbing different parameters with different distributions, and using non-Gaussian distributions. Both of them showed real improvement on highly nonlinear problems (the cart-pole example used a very high-dimensional policy), without knowledge of the problem's dynamics and structure. We believe that SNR-optimized PG algorithms show promise for many complicated, real-world applications.

## 6 Acknowledgements

The authors thank Drs. Lionel Moret and Jun Zhang for valuable assistance with the heaving foil.

## References

Amari, S. (1998). Natural gradient works efficiently in learning. *Neural Computation*, *10*, 251–276.

Baxter, J., & Bartlett, P. (2001). Infinite-horizon policy-gradient estimation. *Journal of Artificial Intelligence Research*, *15*, 319–350.

Greensmith, E., Bartlett, P. L., & Baxter, J. (2004). Variance reduction techniques for gradient estimates in reinforcement learning. *Journal of Machine Learning Research*, *5*, 1471–1530.

Jabri, M., & Flower, B. (1992). Weight perturbation: An optimal architecture and learning technique for analog VLSI feedforward and recurrent multilayer networks. *IEEE Trans. Neural Netw.*, *3*, 154–157.

Kohl, N., & Stone, P. (2004). Policy gradient reinforcement learning for fast quadrupedal locomotion. *Proceedings of the IEEE International Conference on Robotics and Automation (ICRA)*.

Meuleau, N., Peshkin, L., Kaelbling, L. P., & Kim, K.-E. (2000). Off-policy policy search. *NIPS*.

Peters, J., Vijayakumar, S., & Schaal, S. (2003a). *Policy gradient methods for robot control* (Technical Report CS-03-787). University of Southern California.

Peters, J., Vijayakumar, S., & Schaal, S. (2003b). Reinforcement learning for humanoid robotics. *Proceedings of the Third IEEE-RAS International Conference on Humanoid Robots*.

Riedmiller, M., Peters, J., & Schaal, S. (2007). Evaluation of policy gradient methods and variants on the cart-pole benchmark. *Symposium on Approximate Dynamic Programming and Reinforcement Learning* (pp. 254–261).

Shelley, M., Vandenberghe, N., & Zhang, J. (2005). Heavy flags undergo spontaneous oscillations in flowing water. *Physical Review Letters*, *94*.

Tedrake, R., Zhang, T. W., & Seung, H. S. (2004). Stochastic policy gradient reinforcement learning on a simple 3D biped. *Proceedings of the IEEE International Conference on Intelligent Robots and Systems (IROS)* (pp. 2849–2854). Sendai, Japan.

Vandenberghe, N., Zhang, J., & Childress, S. (2004). Symmetry breaking leads to forward flapping flight. *Journal of Fluid Mechanics*, *506*, 147–155.

Williams, J. L., III, J. W. F., & Willsky, A. S. (2006). Importance sampling actor-critic algorithms. *Proceedings of the 2006 American Control Conference*.

Williams, R. (1992). Simple statistical gradient-following algorithms for connectionist reinforcement learning. *Machine Learning*, *8*, 229–256.

